# Bayesian Detection of Infrequent Differences in Sets of Time Series with Shared Structure

**Jennifer Listgarten**[†]**, Radford M. Neal**[†]**, Sam T. Roweis**[†] **Rachel Puckrin**[‡] **and Sean Cutler**[‡]
[†] Department of Computer Science, [‡] Department of Botany,
University of Toronto, Toronto, Ontario, M5S 3G4
{jenn,radford,roweis}@cs.toronto.edu
rachel_puckrin@hotmail.com, cutler@botany.utoronto.ca

## Abstract

We present a hierarchical Bayesian model for sets of related, but different, classes of time series data. Our model performs alignment simultaneously across all classes, while detecting and characterizing class-specific differences. During inference the model produces, for each class, a distribution over a canonical representation of the class. These class-specific canonical representations are automatically aligned to one another — preserving common sub-structures, and highlighting differences. We apply our model to compare and contrast solenoid valve current data, and also, liquid-chromatography-ultraviolet-diode array data from a study of the plant *Arabidopsis thaliana*.

## 1 Aligning Time Series From Different Classes

Many practical problems over a wide range of domains require synthesizing information from several noisy examples of one or more categories in order to build a model which captures common structure and also learns the patterns of variability between categories. In time series analysis, these modeling goals manifest themselves in the tasks of *alignment* and *difference detection*. These tasks have diverse applicability, spanning speech & music processing, equipment & industrial plant diagnosis/monitoring, and analysis of biological time series such as microarray & liquid/gas chromatography-based laboratory data (including mass spectrometry and ultraviolet diode arrays).

Although alignment and difference detection have been extensively studied as separate problems in the signal processing and statistical pattern recognition communities, to our knowledge, no existing model performs both tasks in a unified way. *Single class alignment* algorithms attempt to align a set of time series all together, assuming that variability across different time series is attributable purely to noise. In many real-world situations, however, we have time series from multiple classes (categories) and our prior belief is that there is both substantial shared structure between the class distributions and, simultaneously, systematic (although often rare) differences between them. While in some circumstances (if differences are small and infrequent), single class alignment can be applied to multi-class data, it is much more desirable to have a model which performs true *multi-class alignment* in a principled way, allowing for more refined and accurate modeling of the data.

In this paper, we introduce a novel hierarchical Bayesian model which simultaneously solves the multi-class alignment and difference detection tasks in a unified manner, as illustrated in Figure 1. The single-class alignment shown in this figure coerces the feature in region A for class 1 to be inappropriately collapsed in time, and the overall width of the main broad peak in class 2 to be inappropriately narrowed. In contrast, our multi-class model handles these features correctly. Furthermore, because our algorithm does inference for a fully probabilistic model, we are able to obtain quantitative measures of the posterior uncertainty in our results, which, unlike the point estimates produced by most current approaches, allow us to assess our relative confidence in differences learned by the model. Our basic setup for multi-class alignment assumes the class labels are known

for each time series, as is the case for most difference detection problems. However, as we discuss at the end of the paper, our model can be extended to the completely unsupervised case.

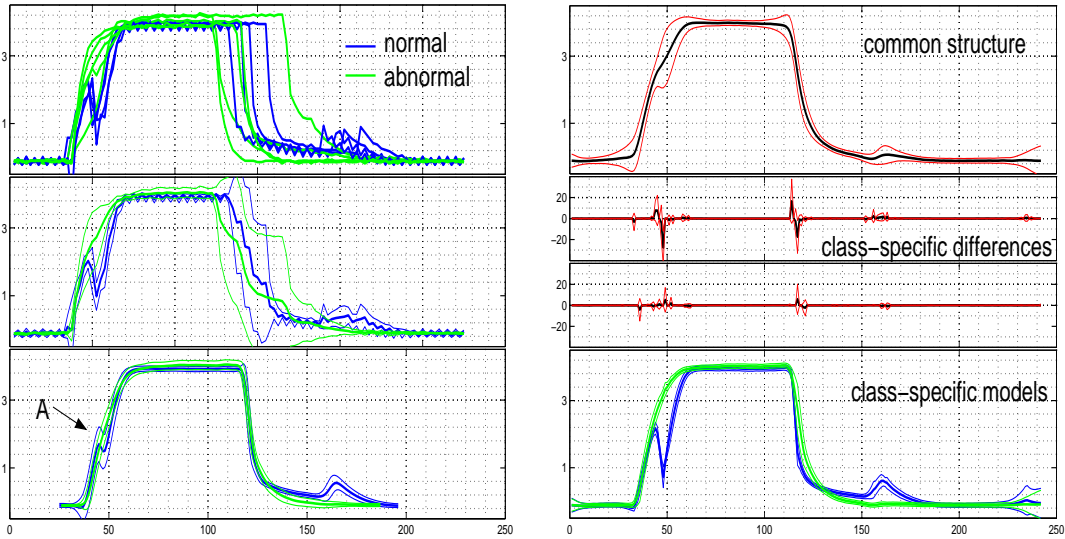

Figure 1: Nine time series from the NASA valve solenoid current data set [4]. Four belong to a 'normal' class, and five to an 'abnormal' class. On all figures, the horizontal axis is time, or latent time for figures of latent traces and observed time series aligned to latent traces. The vertical axis is current amplitude. Top left: The raw, unaligned data. Middle left: Average of the unaligned data within each class in thick line, with the thin lines showing one standard deviation on either side. Bottom left: Average of the aligned data (over MCMC samples) within each class, using the single-class alignment version of the model (no child traces), again with one standard deviation lines shown in the thinner style line. Right: Mean and one standard deviation over MCMC samples using the HB-CPM. Top right: Parent trace. Middle right: Class-specific energy impulses with the top-most showing the class impulses for the less smooth class. Bottom right: Child traces superimposed. Note that if one generates more HB-CPM MCMC samples, the parent cycles between the two classes since the model has no preference for which class is seen as a modification of the other; the child classes remain stable however.

## 2 A Hierarchical Bayesian Continuous Profile Model

Building on our previous Continuous Profile Model (CPM) [7], we propose a Hierarchical Bayesian Continuous Profile Model (HB-CPM) to address the problems of multi-class alignment and difference detection, together, for sets of sibling time series data — that is, replicate time series from several distinct, but related classes. The HB-CPM is a generative model that allows simultaneous alignment of time series and also provides aligned canonical representations of each class along with measures of uncertainty on these representations. Inference in the model can be used, for example, to detect and quantify similarities and differences in class composition. The HB-CPM extends the basic CPM in two significant ways: i) it addresses the multi-class rather than the single-class alignment problem, and ii) it uses a fully Bayesian framework rather than a maximum likelihood approach, allowing us to estimate uncertainty in both the alignments and the canonical representations.

Our model, depicted in Figure 2, assumes that each observed time series is generated as a noisy transformation of a single, class-specific latent trace. Each latent trace is an underlying, noiseless representation of the set of replicated, observable time series belonging to a single class. An observed time series is generated from this latent trace exactly as in the original CPM, by moving through a sequence of hidden states in a Markovian manner and emitting an observable value at each step, as with an HMM. Each hidden state corresponds to a 'latent time' in the latent trace. Thus different choices of hidden state sequences result in different nonlinear transformations of the underlying trace. The HB-CPM uses a separate latent trace for each class, which we call *child traces*. Crucially, each of these child traces is generated from a single *parent trace* (also unobserved), which

captures the common structure among all of the classes. The joint prior distribution for the child traces in the HB-CPM model can be realized by first sampling a parent trace, and then, for each class, sampling a sparse 'difference vector' which dictates how and where each child trace should differ from the common parent.

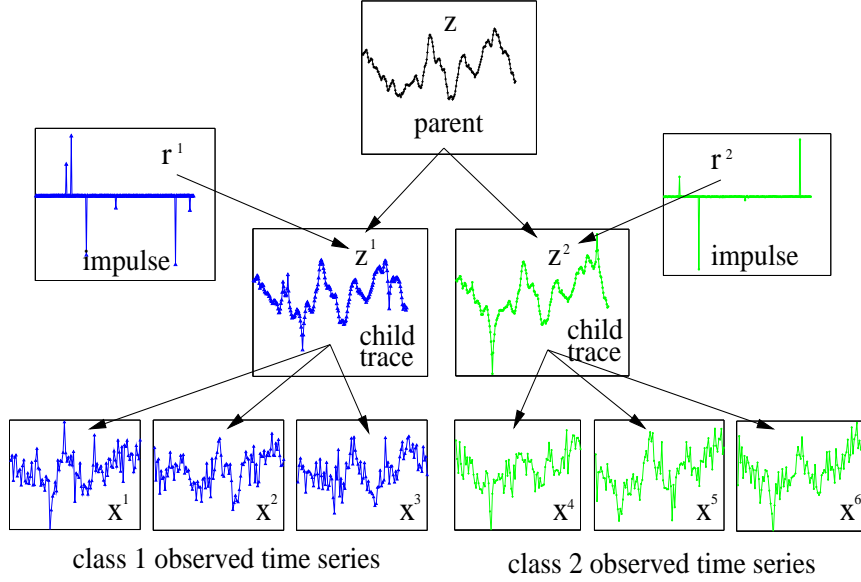

Figure 2: Core elements of the HB-CPM, illustrated with two-class data (hidden and observed) drawn from the model's prior.

class 1 observed time series

class 2 observed time series

## 2.1 The Prior on Latent Traces

Let the vector $\boldsymbol{x}^k = (x_1^k, x_2^k, ..., x_N^k)$ represent the $k^{\text{th}}$ observed scalar time series, and $w^k \in 1..C$ be the class label of this time series. Also, let $\boldsymbol{z} = (z_1, z_2, ..., z_M)$ be the parent trace, and $\boldsymbol{z}^c = (z_1^c, z_2^c, ..., z_M^c)$ be the child trace for the $c^{\text{th}}$ class. During inference, posterior samples of $\boldsymbol{z}^c$ form a canonical representation of the observed times series in class $c$, and $\boldsymbol{z}$ contains their common sub-structure. Ideally, the length of the latent traces, $M$, would be very large relative to $N$ so that any experimental data could be mapped precisely to the correct underlying trace point. Aside from the computational impracticalities this would pose, great care to avoid overfitting would have to be taken. Thus in practice, we have used $M = (2 + \epsilon)N$ (double the resolution, plus some slack on each end) in our experiments, and found this to be sufficient with $\epsilon < 0.2$. Because the resolution of the latent traces is higher than that of the observed time series, experimental time can be made to effectively speed up or slow down by advancing along the latent trace in larger or smaller jumps.

As mentioned previously, the child traces in the HB-CPM inherit most of their structure from a common parent trace. The differences between child and parent are encoded in a difference vector for each class, $\boldsymbol{d}^c = (d_1^c, d_2^c, ..., d_M^c)$; normally, most elements of $\boldsymbol{d}^c$ are close to zero. Child traces are obtained by adding this difference vector to the parent trace: $\boldsymbol{z}^c = \boldsymbol{z} + \boldsymbol{d}^c$.

We model both the parent trace and class-specific difference vectors with what we call an *energy impulse chain*, which is an undirected Markov chain in which neighbouring nodes are encouraged to be similar (*i.e.*, smooth), and where this smoothness is perturbed by a set of marginally independent energy impulse nodes, with one energy impulse node attached to each node in the chain. For the difference vector of the $c^{\text{th}}$ class, the corresponding energy impulses are denoted $\boldsymbol{r}^c = (r_1^c, r_2^c, ..., r_M^c)$, and for the parent trace the energy impulses are denoted $\boldsymbol{r} = (r_1, r_2, ..., r_M)$. Conditioned on the energy impulses, the probability of a difference vector is

$$p(\boldsymbol{d}^c | \boldsymbol{r}^c, \alpha^c, \rho^c) = \frac{1}{Z_{\boldsymbol{r}^c}} \left[ \exp\left( -\frac{1}{2} \left( \sum_{i=1}^{M-1} \frac{(d_i^c - d_{i+1}^c)^2}{\alpha^c} + \sum_{i=1}^{M} \frac{(d_i^c - r_i^c)^2}{\rho^c} \right) \right) \right]. \quad (1)$$

Here, $Z_{\boldsymbol{r}^c}$ is the normalizing constant for this probability density, $\alpha^c$ controls the smoothness of the chain, and $\rho^c$ controls the influence of the energy impulses. Together, $\alpha^c$ and $\rho^c$ also control the overall tightness of the distribution for $\boldsymbol{d}^c$. Presently, we set all $\alpha^c = \alpha'$, and similarly $\rho^c = \rho'$ — that is, these do not differ between classes. Similarly, the conditional probability of the parent trace

is

$$p(\boldsymbol{z}|\boldsymbol{r},\alpha,\rho) = \frac{1}{Z_{\boldsymbol{r}}}\left[\exp\left(-\frac{1}{2}\left(\sum_{i=1}^{M-1}\frac{(z_i-z_{i+1})^2}{\alpha}+\sum_{i=1}^{M}\frac{(z_i-r_i)^2}{\rho}\right)\right)\right]. \qquad (2)$$

These probability densities are each multivariate Gaussian with tridiagonal precision matrixes (corresponding to the Markov nature of the interactions).

Each component of each energy impulse for the parent, $r_j$, is drawn independently from a single univariate Gaussian, $\mathcal{N}(r_i|\mu_{\mathrm{par}},s_{\mathrm{par}})$, whose mean and variance are in turn drawn from a Gaussian and inverse-gamma, respectively. The class-specific difference vector impulses, however, are drawn from a mixture of two zero-mean Gaussians — one 'no difference' (inlier) Gaussian, and one 'class-difference' (outlier) Gaussian. The means are zero so as to encourage difference vectors to be near zero (and thus child traces to be similar to the parent trace). Letting $\delta_i^c$ denote the binary latent mixture component indicator variables for each $r_j^c$,

$$p(\delta_j^c) = \mathrm{Multinomial}(\delta_j^c|m_{\mathrm{in}}^c, m_{\mathrm{out}}^c) = (m_{\mathrm{in}}^c)^{\delta_j^c}(m_{\mathrm{out}}^c)^{1-\delta_j^c} \qquad (3)$$

$$p(r_j^c|\delta_j^c) = \begin{cases} \mathcal{N}(r_j^c|0,s_{\mathrm{in}}^2), & \text{if } \delta_j^c = 1 \\ \mathcal{N}(r_j^c|0,s_{\mathrm{out}}^2), & \text{if } \delta_j^c = 0 \end{cases}. \qquad (4)$$

Each Gaussian mixture variance has an Inverse-Gamma prior, which for the 'no difference' variance, $s_{\mathrm{in}}^2$, is set to have very low mean (and not overly dispersed) so that 'no difference' regions truly have little difference from the parent class, while for the 'class-difference' variance, $s_{\mathrm{out}}^2$, the prior is set to have a larger mean, so as to model our belief that substantial class-specific differences do occasionally exist. The priors for $\alpha^c$, $\rho^c$, $\alpha$, $\rho$ are each log-normal (inverse-gamma priors would not be conjugate in this model, so we use log-normals which are easier to specify). Additionally, the mixing proportions, $m_{\mathrm{in}}^c, m_{\mathrm{out}}^c$, have a Dirichlet prior, which typically encodes our belief that the proportion that are 'class differences' is likely to be small.

## 2.2 The HMM Portion of the Model

Each observed $\boldsymbol{x}^k$ is modeled as being generated by an HMM conditioned on the appropriate child trace, $\boldsymbol{z}^{w^k}$. The probability of an observed time series conditioned on a path of hidden time states, $\boldsymbol{\tau}^k$, and the child trace, is given by $p(\boldsymbol{x}^k|\boldsymbol{z}^{w^k},\boldsymbol{\tau}^k) = \prod_{i=1}^{N}\mathcal{N}(x_i^k|z_{\tau_i^k}^{w^k}u^k,\xi^k)$, where $\xi^k$ is the emission variance for time series $k$, and the scale factor, $u^k$, allows for constant, global, multiplicative rescaling. The HMM transition probabilities $T^k(\tau_{i-1}^k \to \tau_i^k)$ are multinomial within a limited range, with $p^k(\tau_i = a|\tau_{i-1} = b) = \kappa_{(a-b)}^k$ for $(a-b) \in [1, J_\tau]$ and $p^k(\tau_i = a|\tau_{i-1} = b) = 0$ for $(a-b) < 1$ or $(a-b) > J_\tau$ where $J_\tau$ is the maximum allowable number of consecutive time states that can be advanced in a single transition. (Of course, $\sum_{i=1}^{J_\tau}\kappa_i^k = 1$.) This multinomial distribution, in turn, has a Dirichlet prior. The HMM emission variances, $\xi^k$, have an inverse-gamma prior. Additionally, the prior over the first hidden time state is a uniform distribution over a constant number of states, $1..Q$, where $Q$ defines how large a shift can exist between any two observed time series. The prior over each global scaling parameter, $u^k$, is a log-normal with fixed variance and mean of zero, which encourages the scaling factors to remain near unity.

## 3 Posterior Inference of Alignments and Parameters by MCMC

Given a set of observed time series (and their associated class labels), the main computational operation to be performed in the HB-CPM is inference of the latent traces, alignment state paths and other model parameters. Exact inference is analytically intractable, but we are able to use Markov Chain Monte Carlo (MCMC) methods to create an iterative algorithm which, if run for sufficiently long, produces samples from the correct posterior distribution. This posterior provides simultaneous alignments of all observed time series in all classes, and also, crucially, *aligned canonical representations of each class, along with error bars on these representations*, allowing for a principled approach to difference detection in time series data from different classes.

We may also wish to obtain a posterior estimate of some of our parameters conditioned on the data, and marginalized over the other parameters. In particular, we might be interested in obtaining the

posterior over hidden time state vectors for each time series, $\boldsymbol{\tau}^k$, which together provide a simultaneous, multi-class alignment of our data. We may, in addition, or, alternatively, be interested in the posterior of the child traces, $\boldsymbol{z}^c$, which together characterize how the classes agree and disagree. The former may be more of interest for visualizing aligned observed time series, or in expanding out aligned scalar time series to a related vector time series, while the latter would be more of interest when looking to characterize differences in multi-class, scalar time series data.

We group our parameters into blocks, and sample these blocks conditioned on the values of the other parameters (as in Gibbs sampling) — however, when certain conditional distributions are not amenable to direct sampling, we use slice sampling [8]. The scalar conditional distributions for each of $\mu_{\mathrm{par}}, s_{\mathrm{par}}, m_{\mathrm{in}}^c, m_{\mathrm{out}}^c, \delta_j^c, \kappa_i^k$ are known distributions, amenable to direct sampling. The conditional distributions for the scalars $\alpha^c, \rho^c, \alpha, \rho$ and $u^k$ are not tractable, and for each of these we use slice sampling (doubling out and shrinking).

The conditional distribution for each of $\boldsymbol{r}$ and $\boldsymbol{r}^c$ is multivariate Gaussian, and we sample directly from each using a Cholesky decomposition of the covariance matrix.

$$p(\boldsymbol{r}|\boldsymbol{z}, \alpha, \rho) = \frac{1}{Z} p(\boldsymbol{z}|\boldsymbol{r}, \alpha, \rho) p(\boldsymbol{r}) = \mathcal{N}(\boldsymbol{r}|\boldsymbol{c}, \boldsymbol{C}) \tag{5}$$

$$p(\boldsymbol{r}^c|\boldsymbol{d}^c, \alpha^c, \rho^c) = \frac{1}{Z} p(\boldsymbol{d}^c|\boldsymbol{r}, \alpha^c, \rho^c) p(\boldsymbol{r}) = \mathcal{N}(\boldsymbol{r}^c|\boldsymbol{b}, \boldsymbol{B}), \tag{6}$$

where, using $\boldsymbol{I}$ to denote the identity matrix,

$$\boldsymbol{C} = \left(\frac{\boldsymbol{S}}{\rho^2} + \boldsymbol{I} s_{\mathrm{par}}{}^{-1}\right)^{-1}, \qquad \boldsymbol{c} = \boldsymbol{C}\left(\frac{\boldsymbol{z}}{\rho} + \boldsymbol{I}\frac{\mu_{\mathrm{par}}}{s_{\mathrm{par}}}\right) \tag{7}$$

$$\boldsymbol{B} = \left(\frac{\boldsymbol{S}^\dagger}{(\rho^c)^2} + \boldsymbol{v}^{c-1}\right)^{-1}, \qquad \boldsymbol{b} = \boldsymbol{B}\frac{\boldsymbol{d}^c}{\rho^c}. \tag{8}$$

The diagonal matrix $\boldsymbol{v}^c$ consists of mixture component variances ($s_{\mathrm{in}}^2$ or $s_{\mathrm{out}}^2$). $\boldsymbol{S}^{-1}$ [or $\boldsymbol{S}^{\dagger-1}$] is the tridiagonal precision matrix of the multivariate normal distribution $p(\boldsymbol{z}|\boldsymbol{r}, \alpha, \rho)$ [or $p(\boldsymbol{d}^c|\boldsymbol{r}^c, \alpha^c, \rho^c)$], and has entries $S_{j,j}^{-1} = \frac{2}{\alpha} + \frac{1}{\rho}$ for $j = 2..(M-1)$, $S_{j,j}^{-1} = \frac{1}{\alpha} + \frac{1}{\rho}$ for $j = 1, M$, and $S_{j,j+1}^{-1} = S_{j+1,j}^{-1} = -\frac{1}{\alpha}$ [or analogously for $\boldsymbol{S}^{\dagger-1}$]. The computation of $\boldsymbol{C}$ and $\boldsymbol{B}$ can be made more efficient by using the Sherman-Morrison-Woodbury matrix inversion lemma. For example, $\boldsymbol{B} = \frac{1}{(\rho^c)^2}(\boldsymbol{S}^{\dagger-1} - \boldsymbol{S}^{\dagger-1}(\boldsymbol{v}^c + \boldsymbol{S}^{\dagger-1})^{-1}\boldsymbol{S}^{\dagger-1})$, and we have $\boldsymbol{S}^{-1}$ [or $\boldsymbol{S}^{\dagger-1}$] almost for free, and no longer need to invert $\boldsymbol{S}$ [or $\boldsymbol{S}^\dagger$] to obtain it.

The conditional distributions of each of $\boldsymbol{z}, \boldsymbol{z}^c$ are also multivariate Gaussians. However, because of the underlying Markov dependencies, their precision matrixes are tridiagonal, and hence we can use belief propagation, in the style of Kalman filtering, followed by a stochastic traceback to sample from them efficiently. Thus each can be sampled in time proportional to $M$ rather than $M^3$, as required for a general multivariate Gaussian.

Lastly, to sample from the conditional distribution of the hidden time vectors for each sample, $\boldsymbol{\tau}^k$, we run belief propagation (analogous to the HMM forward-backward algorithm) followed by a stochastic traceback.

In our experiments, the parent trace was initialized by averaging one smoothed example from each class. The child traces were initialized to the initial parent trace. The HMM states were initialized by a Viterbi decoding with respect to the initial values of the other parameters. The scaling factors were initialized to unity, and the child energy impulses to zero. MCMC was run for 5000 iterations, with convergence generally realized in less than 1000 iterations.

## 4   Experiments and Results

We demonstrate use of the HB-CPM on two data sets. The first data set is the part of the NASA shuttle valve data [4], which measures valve solenoid current against time for some 'normal' runs and some 'abnormal' runs. Measurements were taken at a rate of 1ms per sample, with 1000 samples per time series. We subsampled the data by a factor of 7 in time since it was extremely dense. The results of performing posterior inference in our model on this two-class data set are shown in

Figure 1. They nicely match our intuition of what makes a good solution. In our experiments, we also compared our model to a simple "single-class" version of the HB-CPM in which we simply remove the child trace level of the model, letting all observed data in both classes depend directly on one single parent trace. The single-class alignment, while doing a reasonable job, does so by coercing the two classes to look more similar than they should. This is evident in one particular region labeled on the graph and discussed in the legend. Essentially a single class alignment causes us to lose class-specific fine detail — the precise information we seek to retain for difference detection.

The second data set is from a botany study which uses reverse-phase HPLC (high performance liquid chromatography) as a high-throughput screening method to identify genes involved in xenobiotic uptake and metabolism in the model plant *Arabidopsis thaliana*. Liquid-chromatography (LC) techniques are currently being developed and refined with the aim of providing a robust platform with which to detect differences in biological organisms — be they plants, animals or humans. Detected differences can reveal new fundamental biological insight, or can be applied in more clinical settings. LC-mass spectrometry technology has recently undergone explosive growth in tackling the problem of biomarker discovery — for example, detecting biological markers that can predict treatment outcome or severity of disease, thereby providing the potential for improved health care and better understanding of the mechanisms of drug and disease. In botany, LC-UV data is used to help understand the uptake and metabolism of compounds in plants by looking for differences across experimental conditions, and it is this type of data that we use here.

LC separates mixtures of analytes on the basis of some chemical property — hydrophobicity, for reverse-phase LC, used to generate our data. Components of the analyte in our data set were detected as they came off the LC column with a Diode Array Detector (DAD), yielding UV-visible spectra collected at 540 time points (we used the 280 nm band, which is informative for these experiments). We performed background subtraction [2] and then subsampled this data by a factor of four. This is a three-class data set, where the first class is untreated plant extract, followed by two classes consisting of this same plant treated with compounds that were identified as possessing robust uptake *in vivo*, and, hence, when metabolized, provide a differential LC-UV signal of interest.

Figure 3 gives an overview of the LC-UV results, while Figure 4 zooms in on a particular area of interest to highlight how subtle differences can be detected by the HB-CPM, but not by a single-class alignment scheme. As with the NASA data set, a single-class alignment coerces features across classes that are in fact different to look the same, thereby preventing us from detecting them. Recall that this data set consists of a 'no treatment' plant extract, and two 'treatments' of this same plant. Though our model was not informed of these special relationships, it nevertheless elegantly captures this structure by giving almost no energy impulses to the 'no treatment' class, meaning that this class is essentially the parent trace, and allowing the 'treatment' classes to diverge from it, thereby nicely matching the reality of the situation.

All averaging over MCMC runs shown is over 4000 samples, after a 1000 burn in period, which took around 3 hours for the NASA data, and 5 hours for the LC data set, on machines with dual 3 GHz Pentium 4 processors.

## 5 Related Work

While much work has been done on time series alignment, and on comparison/clustering of time series, none of this work, to our knowledge, directly addresses the problem presented in this paper — simultaneously aligning and comparing sets of related time series in order to characterize how they differ from one another.

The classical algorithm for aligning time series is Dynamic Time Warping (DTW) [10]. DTW works on pairs of time series, aligning one time series to a specified reference time, in a non-probabilistic way, without explicit allowance for differences in related time series. More recently, Gaffney *et al* [5] jointly clustered and aligned time series data from different classes. However, their model does not attempt to put time series from different classes into correspondence with one another — only time series within a class are aligned to one another. Ziv Bar-Joseph *et al* [1] use a similar approach to cluster and align microarray time series data. Ramsay *et al* [9] have introduced a curve clustering model, in which a time warping function, $h(t)$, for each time series is learned by way of learning its relative curvature, parameterized with order one B-spline coefficients. This model accounts for

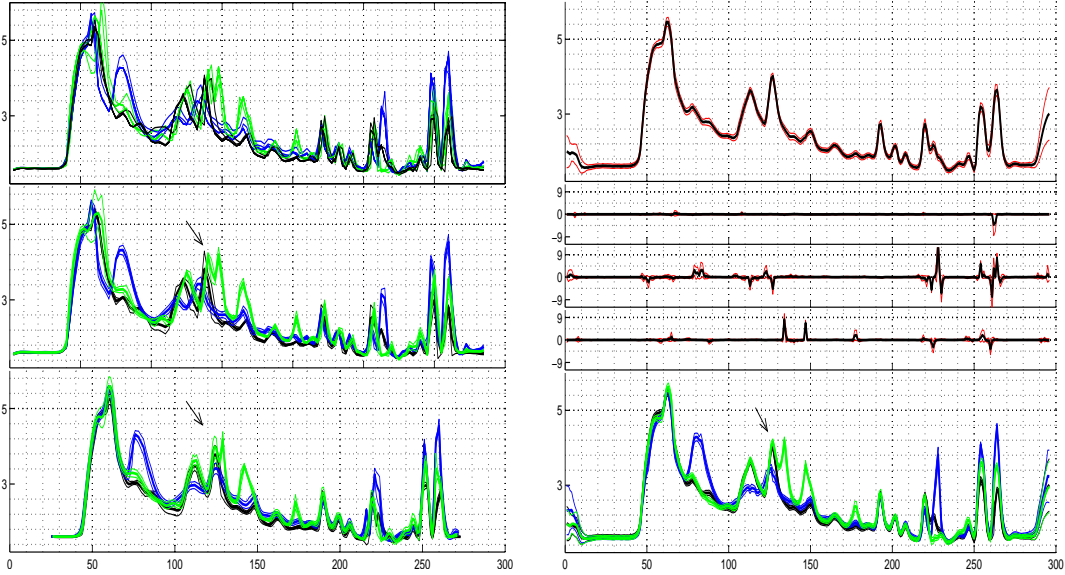

Figure 3: Seven time series from each of three classes of LC-UV data. On all figures, the horizontal axis is time, or latent time for figures of latent traces and observed time series aligned to latent traces. The vertical axis is log of UV absorbance. Top left: The raw, unaligned data. Middle left: Average of the unaligned data within each class in thick line, with the thin lines showing one standard deviation on either side. Bottom left: Average of the aligned data within each class, using the single-class alignment version of the model (no child traces), again with one standard deviation lines shown in the thinner style line. Right: Mean and one standard deviation over MCMC samples using the HB-CPM model. Top right: Parent trace. Middle right: Class-specific energy impulses, with the top-most showing the class impulses for the 'no treatment' class. Bottom right: Child traces superimposed. See Figure 4 for a zoom-in in around the arrow.

systematic changes in the range and domain of time series in a way that aligns curves with the same fundamental shape. However, their method does not allow for class-specific differences between shapes to be taken into account. The anomaly detection (AD) literature deals with related, yet distinct problems. For example, Chan *et al* [3] build a model of one class of time series data (they use the same NASA valve data as in this paper), and then match test data, possibly belonging to another class (*e.g.* 'abnormal' shuttle valve data) to this model to obtain an anomaly score. Emphasis in the AD community is on detecting abnormal events relative to a normal baseline, in an on-line manner, rather than comparing and contrasting two or more classes from a dataset containing examples of all classes. The problem of 'elastic curve matching' is addressed in [6], where a target time series that best matches a query series is found, by mapping the problem of finding the best matching subsequence to the problem of finding the cheapest path in a DAG (directed acyclic graph).

## 6  Discussion and Conclusion

We have introduced a hierarchical, Bayesian model to perform detection of rare differences between sets of related time series, a problem which arises across a wide range of domains. By training our model, we obtain the posterior distribution over a set of class-specific canonical representations of each class, which are aligned in a way that preserves their common sub-structures, yet retains and highlights important differences.

This model can be extended in several interesting and useful ways. One small modification could be useful for the LC-UV data set presented in this paper, in which one of the classes was 'no treatment', while the other two were each a different 'treatment'. We might model the 'no treatment' as the parent trace, and each of the treatments as a child trace, so that the direct comparison of interest would be made more explicit. Another direction would be to apply the HB-CPM in a completely

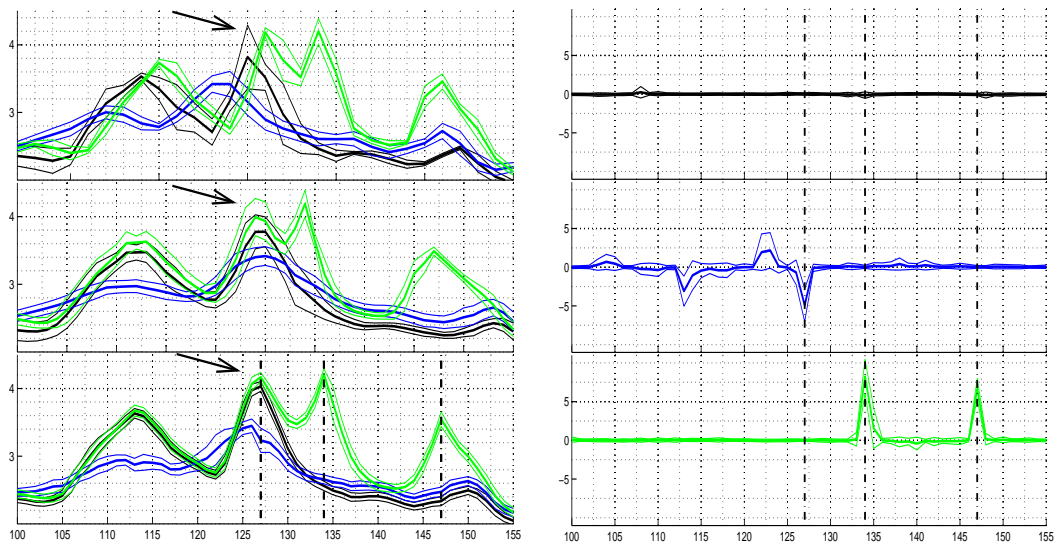

Figure 4: Left: A zoom in of data displayed in Figure 3, from the region of time 100-150 (labeled in that figure in latent time, not observed time). Top left: mean and standard deviation of the unaligned data. Middle left: mean and standard deviation of the single-class alignment. Bottom left: mean and standard deviation of the child traces from the HB-CPM. A case in point of a difference that could be detected with the HB-CPM and not in the raw or single-class aligned data, is the difference occurring at time point 127. Right: The mean and standard deviation of the child energy impulses, with dashed lines showing correspondences with the child traces in the bottom left panel.

unsupervised setting where we learn not only the canonical class representations, but also obtain the posterior over the class labels by introducing a latent class indicator variable. Lastly, one could use a model with cyclical latent traces to model cyclic data such as electrocardiogram (ECG) and climate data. In such a model, an observed trace being generated by the model would be allowed to cycle back to the start of the latent trace, and the smoothness constraints on the trace would be extended to apply to beginning and end of the traces, coercing these to be similar. Such a model would allow one to do anomaly detection in cyclic data, as well as segmentation.

**Acknowledgments**: Thanks to David Ross and Roland Memisevic for useful discussions, and Ben Marlin for his Matlab slice sampling code.

## References

[1] Z. Bar-Joseph, G. Gerber, D. K. Gifford, T. Jaakkola, and I. Simon. A new approach to analyzing gene expression time series data. In *RECOMB*, pages 39–48, 2002.

[2] H. Boelens, R. Dijkstra, P. Eilers, F. Fitzpatrick, and J. Westerhuis. New background correction method for liquid chromatography with diode array detection, infrared spectroscopic detection and raman spectroscopic detection. *Journal of Chromatography A*, 1057:21–30, 2004.

[3] P. K. Chan and M. V. Mahoney. Modeling multiple time series for anomaly detection. In *ICDM*, 2005.

[4] B. Ferrell and S. Santuro. NASA shuttle valve data. `http://www.cs.fit.edu/~pkc/nasa/data/`, 2005.

[5] S. J. Gaffney and P. Smyth. Joint probabilistic curve clustering and alignment. In *Advances in Neural Information Processing Systems 17*, 2005.

[6] L. Latecki, V. Megalooikonomou, Q. Wang, R. Lakaemper, C. Ratanamahatana, and E. Keogh. Elastic partial matching of time series, 2005.

[7] J. Listgarten, R. M. Neal, S. T. Roweis, and A. Emili. Multiple alignment of continuous time series. In *Advances in Neural Information Processing Systems 17*, 2005.

[8] R. M. Neal. Slice sampling. *Annals of Statistics*, 31:705–767, 2003.

[9] J. Ramsay and X. Li. Curve registration. *Journal of the Royal Statistical Society(B)*, 60, 1998.

[10] H. Sakoe and S. Chiba. Dynamic programming algorithm for spoken word recognition. *Readings in Speech Recognition*, pages 159–165, 1990.